# Fast Variational Inference
# for Large-scale Internet Diagnosis

**John C. Platt**          **Emre Kıcıman**          **David A. Maltz**

Microsoft Research
1 Microsoft Way
Redmond, WA 98052
{jplatt,emrek,dmaltz}@microsoft.com

## Abstract

Web servers on the Internet need to maintain high reliability, but the cause of intermittent failures of web transactions is non-obvious. We use approximate Bayesian inference to diagnose problems with web services. This diagnosis problem is far larger than any previously attempted: it requires inference of $10^4$ possible faults from $10^5$ observations. Further, such inference must be performed in less than a second. Inference can be done at this speed by combining a mean-field variational approximation and the use of stochastic gradient descent to optimize a variational cost function. We use this fast inference to diagnose a time series of anomalous HTTP requests taken from a real web service. The inference is fast enough to analyze network logs with billions of entries in a matter of hours.

## 1  Introduction

Internet content providers, such as MSN, Google and Yahoo, all depend on the correct functioning of the wide-area Internet to communicate with their users and provide their services. When these content providers lose network connectivity with some of their users, it is critical that they quickly resolve the problem, even if the failure lies outside their own systems. [1]  One challenge is that content providers have little direct visibility into the wide-area Internet infrastructure and the causes of user request failures. Requests may fail because of problems in the content provider's systems or faults in the network infrastructure anywhere between the user and the content provider, including routers, proxies, firewalls, and DNS servers. Other failing requests may be due to denial of service attacks or bugs in the user's software. To compound the diagnosis problem, these faults may be intermittent: we must use probabilistic inference to perform diagnosis, rather than using logic.

A second challenge is the scale involved. Not only do popular Internet content providers receive billions of HTTP requests a week, but the number of potential causes of failure are numerous. Counting only the coarse-grained Autonomous Systems (ASes) through which users receive Internet connectivity, there are over 20k potential causes of failure. In this paper, we show that approximate Bayesian inference scales to handle this high rate of observations and accurately estimates the underlying failure rates of such a large number of potential causes of failure.

To scale Bayesian inference to Internet-sized problems, we must make several simplifying approximations. First, we introduce a bipartite graphical model using overlapping noisy-ORs, to model the interactions between faults and observations. Second, we use mean-

field variational inference to map the diagnosis problem to a reasonably-sized optimization problem. Third, we further approximate the integral in the variational method. Fourth, we speed up the optimization problem using stochastic gradient descent.

The paper is structured as follows: Section 1.1 discusses related work to this paper. We describe the graphical model in Section 2, and the approximate inference in that model in Section 2.1, including stochastic gradient descent (in Section 3). We present inference results on synthetic and real data in Section 4 and then draw conclusions.

## 1.1  Previous Work

The original application of Bayesian diagnosis was medicine. One of the original diagnosis network was QMR-DT [14], a bipartite graphical model that used noisy-OR to model symptoms given diseases. Exact inference in such networks is intractable (exponential in the number of positive symptoms,[2]), so different approximation and sampling algorithms were proposed. Shwe and Cooper proposed likelihood-weighted sampling [13], while Jaakkola and Jordan proposed using a variational approximation to unlink each input to the network [3]. With only thousands of possible symptoms and hundreds of diseases, QMR-DT was considered very challenging.

More recently, researchers have applied Bayesian techniques for the diagnosis of computers and networks [1][12][16]. This work has tended to avoid inference in large networks, due to speed constraints. In contrast, we attack the enormous inference problem directly.

## 2  Graphical model of diagnosis

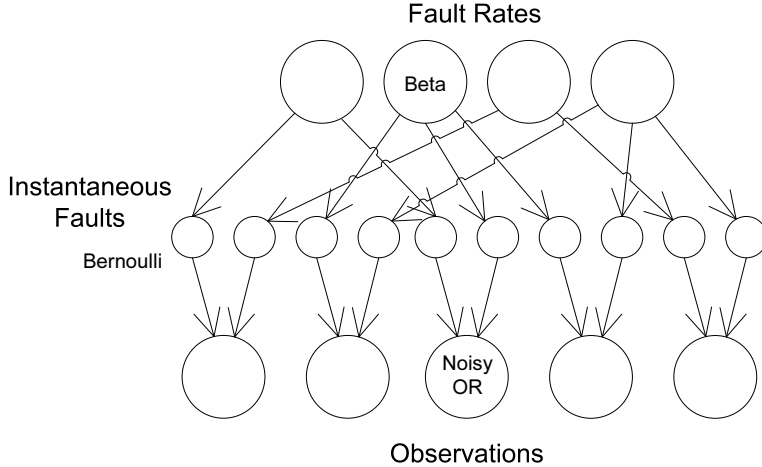

Figure 1: The full graphical model for the diagnosis of Internet faults

The initial graphical model for diagnosis is shown in Figure 1. Starting at the bottom, we observe a large number of binary random variables, each corresponding to the success/failure of a single HTTP request. The failure of an HTTP request can be modeled as a noisy-OR [11] of a set of Bernoulli-distributed binary variables, each of which models the underlying factors that can cause a request to fail:

$$P(V_i = \text{fail}|D_{ij}) = 1 - (1 - r_{i0})\prod_j(1 - r_{ij}d_{ij}), \qquad (1)$$

where $r_{ij}$ is the probability that the observation is a failure if a single underlying fault $d_{ij}$ is present. The matrix $r_{ij}$ is typically very sparse, because there are only a small number of possible causes for the failure of any request. The $r_{i0}$ parameter models the probability of a spontaneous failure without any known cause. The $r_{ij}$ are set by elicitation of probabilities from an expert.

The noisy-OR models the causal structure in the network, and its connections are derivable from the metadata associated with the HTTP request. For example, a single request can fail

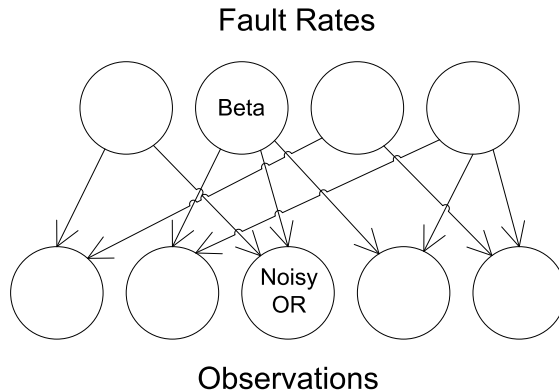

Figure 2: Graphical model after integrating out instantaneous faults: a bipartite noisy-OR network with Beta distributions as hidden variables

because its server has failed, or because a misconfigured or overloaded router can cause an AS to lose connectivity to the content provider, or because the user agent is not compatible with the service. All of these underlying causes are modeled independently for each request, because possible faults in the system can be intermittent.

Each of the Bernoulli variables $D_{ij}$ depends on an underlying continuous fault rate variable $F_j \in [0, 1]$:

$$P(D_{ij}|F_j = \mu_j) = \mu_j^{d_{ij}}(1 - \mu_j)^{1-d_{ij}}, \tag{2}$$

where $\mu_j$ is the probability of a fault manifesting at any time. We model the $F_j$ as independent Beta distributions, one for each fault:

$$p(F_j = \mu_j) = \frac{1}{\mathrm{B}(\alpha_j^0, \beta_j^0)} \mu_j^{\alpha_j^0 - 1}(1 - \mu_j)^{\beta_j^0 - 1}, \tag{3}$$

where B is the beta function. The fan-out for each of these fault rates can be different: some of these fault rates are connected to many observations, while less common ones are connected to fewer.

Our goal is to model the posterior distribution $P(\vec{F}|\vec{V})$ in order to identify hidden faults and track them through time. The existence of the $D_{ij}$ random variable is a nuisance. We do not want to estimate $P(\vec{D}|\vec{V})$ for any $D_{ij}$: the distribution of instantaneous problems is not interesting. Fortunately, we can exactly integrate out these nuisance variables, because they are connected to only one observation thru a noisy-OR.

After integrating out the $D_{ij}$, the graphical model is shown in Figure 2. The model is now completely analogous to the QMR-DT mode [14], but instead of the noisy-OR combining binary random variables, they combine rate variables:

$$P(V_i = \mathrm{fail}|F_j = \mu_j) = 1 - (1 - r_{i0})\prod_j(1 - r_{ij}\mu_j). \tag{4}$$

One can view (4) as a generalization of a noisy-OR to continuous $[0, 1]$ variables.

## 2.1 Approximations to make inference tractable

In order to scale inference up to $10^4$ hidden variables, and $10^5$ observations, we choose a simple, robust approximate inference algorithm: mean-field variational inference [4]. Mean-field variational inference approximates the posterior $P(\vec{F}|\vec{V})$ with a factorized distribution. For inferring fault rates, we choose to approximate $P$ with a product of beta distributions

$$Q(\vec{F}|\vec{V}) = \prod_j q(F_j|\vec{V}) = \prod_j \frac{1}{\mathrm{B}(\alpha_j, \beta_j)} \mu_j^{\alpha_j - 1}(1 - \mu_j)^{\beta_j - 1}. \tag{5}$$

Mean-field variational inference maximizes a lower bound on the evidence of the model:

$$\max_{\vec{\alpha},\vec{\beta}} \mathcal{L} = \int Q(\vec{\mu}|\vec{V}) \log \frac{P(\vec{V}|\vec{\mu})p(\vec{\mu})}{Q(\vec{\mu}|\vec{V})} \, \mathrm{d}\vec{\mu}. \tag{6}$$

This integral can be broken into two terms: a cross-entropy between the approximate posterior and the prior, and an expected log-likelihood of the observations:

$$\max_{\vec{\alpha},\vec{\beta}} \mathcal{L} = -\int Q(\vec{\mu}|\vec{V}) \log \frac{Q(\vec{\mu}|\vec{V})}{p(\vec{\mu})} \, d\vec{\mu} + \left\langle \log P(\vec{V}|\vec{F}) \right\rangle_Q. \tag{7}$$

The first integral is the negative of a sum of cross-entropies between Beta distributions with a closed form:

$$
\begin{aligned}
D_{\mathrm{KL}}(q_j || p_j) &= \log\left(\frac{\mathrm{B}(\alpha_j^0, \beta_j^0)}{\mathrm{B}(\alpha_j, \beta_j)}\right) + (\alpha_j - \alpha_j^0)\psi(\alpha_j) \\
&\quad + (\beta_j - \beta_j^0)\psi(\beta_j) - (\alpha_j + \beta_j - \alpha_j^0 - \beta_j^0)\psi(\alpha_j + \beta_j),
\end{aligned}
\tag{8}
$$

where $\psi$ is the digamma function.

However, the expected log likelihood of a noisy-OR integrated over a product of Beta distributions does not have an analytic form. Therefore, we employ the MF(0) approximation of Ng and Jordan [9], replacing the expectation of the log likelihood with the log likelihood of the expectation. The second term then becomes the sum of a set of log likelihoods, one per observation:

$$
L(V_i) = \begin{cases} \log\left(1 - (1-r_{i0})\prod_j[1 - r_{ij}\alpha_j/(\alpha_j + \beta_j)]\right) & \text{if } V_i = 1 \text{ (failure)};\\ \log(1 - r_{i0}) + \sum_j \log[1 - r_{ij}\alpha_j/(\alpha_j + \beta_j)] & \text{if } V_i = 0 \text{ (success)}. \end{cases}
\tag{9}
$$

For the Internet diagnosis case, the MF(0) approximation is reasonable: we expect the posterior distribution to be concentrated around its mean, due to the large amount of data that is available. Ng and Jordan [9] have have proved accuracy bounds for MF(0) based on the number of parents that an observation has.

The final cost function for a minimization routine then becomes

$$\min_{\vec{\alpha},\vec{\beta}} C = \sum_j D_{\mathrm{KL}}(q_j || p_j) - \sum_i L(V_i). \tag{10}$$

## 3  Variational inference by stochastic gradient descent

In order to apply unconstrained optimization algorithms to minimize (10), we need transform the variables: only positive $\alpha_j$ and $\beta_j$ are valid, so we parameterize them by

$$\alpha_j = e^{a_j}, \qquad \beta_j = e^{b_j}. \tag{11}$$

and the gradient computation becomes

$$\frac{\partial C}{\partial a_j} = \alpha_j \left(\sum_j \frac{\partial D_{\mathrm{KL}}(q_j || p_j)}{\partial \alpha_j} - \sum_i \frac{\partial L(V_i)}{\partial \alpha_j}\right). \tag{12}$$

with a similar gradient for $b_j$. Note that this gradient computation can be quite computationally expensive, given that $i$ sums over all of the observations.

For Internet diagnosis, we can decompose the observation stream into blocks, where the size of the block is determined by how quickly the underlying rates of faults change, and how finely we want to sample those rates. We typically use blocks of 100,000 observations, which can make the computation of the gradient expensive. Further, we repeat the inference over and over again, on thousands of blocks of data: we prefer a fast optimization procedure over a highly accurate one.

Therefore, we investigated the use of stochastic gradient descent for optimizing the variational cost function. Stochastic gradient descent approximates the full gradient with a

---
**Algorithm 1** Variational Gradient Descent
---
**Require:** Noisy-OR parameters $r_{ij}$, priors $\alpha_j^0, \beta_j^0$, observations $V_i$
    Initialize $a_j = \log(\alpha_j^0), b_j = \log(\beta_j^0)$
    Initialize $y_i, z_j$ to 0
    **for** $k = 1$ to number of epochs **do**
        **for all** Faults $j$ **do**
            $\alpha_j = \exp(a_j), \beta_j = \exp(b_j)$
            $y_j \leftarrow \xi y_j + (1 - \xi)\partial D_{\mathrm{KL}}(q_j \| p_j; \alpha_j, \beta_j)/\partial a_j$
            $z_j \leftarrow \xi z_j + (1 - \xi)\partial D_{\mathrm{KL}}(q_j \| p_j; \alpha_j, \beta_j)/\partial b_j$
            $a_j \leftarrow a_j - \eta y_j$
            $b_j \leftarrow b_j - \eta z_j$
        **end for**
        **for all** Observations $i$ **do**
            **for all** Parent faults $j$ of observation $v_i$ **do**
                $\alpha_j = \exp(a_j), \beta_j = \exp(b_j)$
            **end for**
            **for all** Parent faults $j$ of observation $v_i$ **do**
                $y_j \leftarrow \xi y_j - (1 - \xi)\partial L(V_i; \vec{\alpha}, \vec{\beta})/\partial a_j$
                $z_j \leftarrow \xi z_j - (1 - \xi)\partial L(V_i; \vec{\alpha}, \vec{\beta})/\partial b_j$
                $a_j \leftarrow a_j - \eta y_j$
                $b_j \leftarrow b_j - \eta z_j$
            **end for**
        **end for**
    **end for**
---

single term from the gradient: the state of the optimization is updated using that single term [5]. This enables the system to converge quickly to an approximate answer. The details of stochastic gradient descent are shown in Algorithm 1.

Estimating the sum in equation (12) with a single term adds a tremendous amount of noise to the estimates. For example, the sign of a single $L(V_i)$ gradient term depends only on the sign of $V_i$. In order to reduce the noise in the estimate, we use momentum [15]: we exponentially smooth the gradient with a first-order filter before applying it to the state variables. This momentum modification is shown in Algorithm 1. We typically use a large step size ($\eta = 0.1$) and momentum term ($\xi = 0.99$), in order to both react quickly to changes in the fault rate and to smooth out noise.

Stochastic gradient descent can be used as a purely on-line method (where each data point is seen only once), setting the "number of epochs" in Algorithm 1 to 1. Alternatively, it can get higher accuracy if it is allowed to sweep through the data multiple times.

## 3.1   Other possible approaches

We considered and tested several other approaches to solving the approximate inference problem.

Jaakkola and Jordan propose a variational inference method for bipartite noisy-OR networks [3], where one variational parameter is introduced to unlink one observation from the network. We typically have far more observations than possible faults: this previous approach would have forced us to solve very large optimization problems (with 100,000 parameters). Instead, we solve an optimization that has dimension equal to the number of faults.

We originally optimized the variational cost function (10) with both BFGS and the trust-region algorithm in the Matlab optimization toolbox. This turned out to be far worse than stochastic gradient descent. We found that a C# implementation of L-BFGS, as described in Nocedal and Wright [10] sped up the exact optimization by orders of magnitude. We report on the L-BFGS performance, below: it is within 4x the speed of the stochastic gradient descent.

We experimented with Metropolis-Hastings to sample from the posterior, using a Gaussian random walk in $(a_j, b_j)$. We found that the burn-in time was very long. Also, each update is slow, because the speed of a single update depends on the fan-out of each fault. In the Internet diagnosis network, the fan-out is quite high (because a single fault affects many observations). Thus, Metropolis-Hastings was far slower than variational inference.

We did not try loopy belief propagation [8], nor expectation propagation [6]. Because the Beta distribution is not conjugate to the noisy OR, the messages passed by either algorithm do not have a closed form.

Finally, we did not try the idea of learning to predict the posterior from the observations by sampling from the generative model and learning the reverse mapping [7]. For Internet diagnosis, we do not know the structure of graphical model for a block of data ahead of time: the structure depends on the metadata for the requests in the log. Thus, we cannot amortize the learning time of a predictive model.

# 4  Results

We test the approximations and optimization methods used for Internet diagnosis on both synthetic and real data.

## 4.1  Synthetic data with known hidden state

Testing the accuracy of approximate inference is very difficult, because, for large graphical models, the true posterior distribution is intractable. However, we can probe the reliability of the model on a synthetic data set.

We start by generating fault rates from a prior (here, 2000 faults drawn from Beta(5e-3,1)). We randomly generate connections from faults to observations, with probability $5 \times 10^{-3}$. Each connection has a strength $r_{ij}$ drawn randomly from $[0, 1]$. We generate 100,000 observations from the noisy-OR model (4). Given these observations, we predict an approximate posterior.

Given that the number of observations is much larger than the number of faults, we expect that the posterior distribution should tightly cluster around the rate that generated the observations. Difference between the true rate and the mean of the approximate posterior should reflect inaccuracies in the estimation.

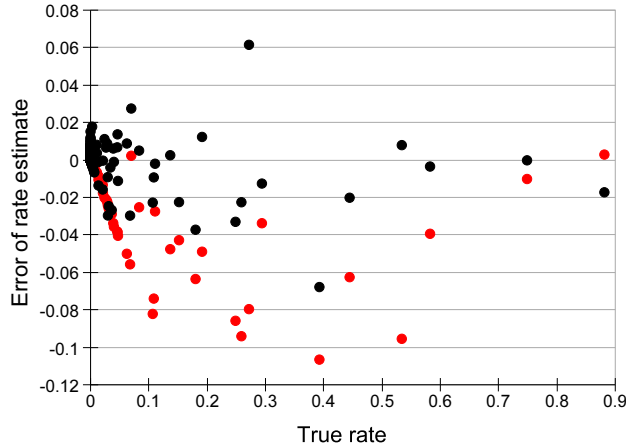

Figure 3: The error in estimate of rate versus true underlying rate. Black dots are L-BFGS, Red dots are Stochastic Gradient Descent with 20 epochs.

The results for a run is shown in Figure 3. The figure shows that the errors in the estimate are small enough to be very useful for understanding network errors. There is a slight systematic bias in the stochastic gradient descent, as compared to L-BFGS. However, the improvement in speed shown in Table 1 is worth the loss of accuracy: we need inference to

be as fast as possible to scale to billions of samples. The run times are for a uniprocessor Pentium 4, 3 GHz, with code in C#.

| Algorithm | Accuracy (RMSE) | Time (CPU sec) |
|---|---|---|
| L-BFGS | 0.0033 | 38 |
| SGD, 1 epoch | 0.0343 | 0.5 |
| SGD, 20 epochs | 0.0075 | 11.7 |

Table 1: Accuracy and speed on synthetic data set

## 4.2 Real data from web server logs

We then tested the algorithm on real data from a major web service. Each observation consists of a success or failure of a single HTTP request. We selected 18848 possible faults that occur frequently in the dataset, including the web server that received the request, which autonomous system that originated the request, and which "user agent" (brower or robot) generated the request.

We have been analyzing HTTP logs collected over several months with the stochastic gradient descent algorithm. In this paper, we present an analysis of a short 2.5 hour window containing an anomalously high rate of failures, in order to demonstrate that our algorithm can help us understand the cause of failures based on observations in a real-world environment.

We broke the time series of observations into blocks of 100,000 observations, and inferred the hidden rates for each block. The initial state of the optimizer was set to be the state of the optimizer at convergence of the previous block. Thus, for stochastic gradient descent, the momentum variables were carried forward from block to block.

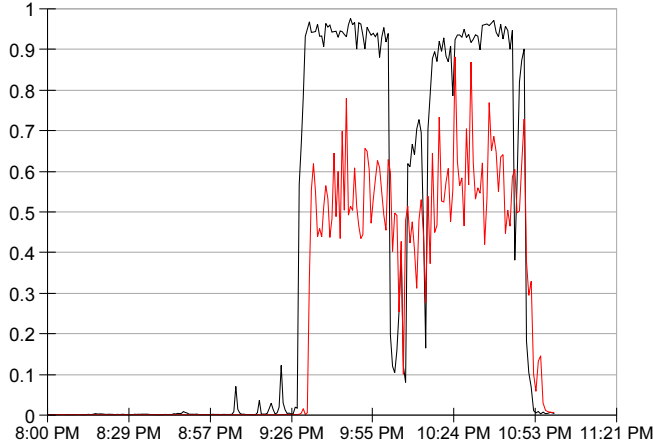

Figure 4: The inferred fault rate for two Autonomous Systems, as a function of time. These are the only two faults with high rate.

The results of this tracking experiment are shown in Figure 4. In this figure, we used stochastic gradient descent and a Beta(0.1,100) prior. The figure shows the only two faults whose probability went higher than 0.1 in this time interval: they correspond to two ASes in the same city, both causing failures at roughly the same time. This could be due to a router that is in common between them, or perhaps an denial of service attack that originated in that city.

The speed of the analysis is much faster than real time. For a data set of 10 million samples, L-BFGS required 209 CPU seconds, while SGD (with 3 passes of data per block) only required 51 seconds. This allows us to go through logs containing billions of entries in a matter of hours.

# 5 Conclusions

This paper presents high-speed variational inference to diagnose problems on the scale of the Internet. Given observations at a web server, the diagnosis can determine whether a web server needs rebooting, whether part of the Internet is broken, or whether the web server is compatible with a browser or user agent.

In order to scale inference up to Internet-sized diagnosis problems, we make several approximations. First, we use mean-field variational inference to approximate the posterior distribution. The expected log likelihood inside of the variational cost function is approximated with the MF(0) approximation. Finally, we use stochastic gradient descent to perform the variational optimization.

We are currently using variational stochastic gradient descent to analyze logs that contain billions of requests. We are not aware of any other applications of variational inference at this scale. Future publications will include conclusions of such analysis, and implications for web services and the Internet at large.

## Footnotes

[1] A loss of connectivity to users translates directly into lost revenue and a sullied reputation for content providers, even if the cause of the problem is a third-party network component.

# References

[1] M. Chen, A. X. Zheng, J. Lloyd, M. I. Jordan, and E. Brewer. Failure diagnosis using decision trees. In *Proc. Int'l. Conf. Autonomic Computing*, pages 36–43, 2004.

[2] D. Heckerman. A tractable inference algorithm for diagnosing multiple diseases. In *Proc. UAI*, pages 163–172, 1989.

[3] T. Jaakkola and M. Jordan. Variational probabilistic inference and the QMR-DT database. *Journal of Artificial Intelligence Research*, 10:291–322, 1999.

[4] M. I. Jordan, Z. Ghahramani, T. S. Jaakkola, and L. K. Saul. An introduction to variational methods for graphical models. *Machine Learning*, 37:183–233, 1999.

[5] H. J. Kushner and G. G. Yin. *Stochastic Approximation and Recursive Algorithms and Applications*. Springer-Verlag, 2003.

[6] T. P. Minka. Expectation propagation for approximate bayesian inference. In *Proc. UAI*, pages 362–369, 2001.

[7] Q. Morris. Recognition networks for approximate inference in BN20 networks. In *Proc. UAI*, pages 370–37, 2001.

[8] K. P. Murphy, Y. Weiss, and M. I. Jordan. Loopy belief propagation for approximate inference: An empirical study. In *Proc. UAI*, pages 467–475, 1999.

[9] A. Y. Ng and M. Jordan. Approximate inference algorithms for two-layer bayesian networks. In *Proc. NIPS*, pages 533–539, 1999.

[10] J. Nocedal and S. J. Wright. *Numerical Optimization*. Springer, 2nd edition, 2006.

[11] J. Pearl. *Probabilistic Reasoning In Intelligent Systems: Networks of Plausible Inference*. Morgan Kaufmann, 1988.

[12] I. Rish, M. Brodie, and S. Ma. Accuracy vs. efficiency tradeoffs in probabilistic diagnosis. In *Proc. AAAI*, pages 560–566, 2001.

[13] M. A. Shwe and G. F. Cooper. An empirical analysis of likelihood-weighting simulation on a large, multiply-connected medical belief network. *Computers and Biomedical Research*, 24(5):453–475, 1991.

[14] M. A. Shwe, B. Middleton, D. E. Heckerman, M. Henrion, E. J. Horvitz, H. P. Lehmann, and G. F. Cooper. Probabilistic diagnosis using a reformulation of the INTERNIST-1/QMR knowledge base. *Methods of Information in Medicine*, 30(4):241–255, 1991.

[15] J. J. Shynk and S. Roy. The LMS algorithm with momentum updating. In *Proc. Intl. Symp. Circuits and Systems*, pages 2651–2654, 1988.

[16] M. Steinder and A. Sethi. End-to-end service failure diagnosis using belief networks. In *Proc. Network Operations and Management Symposium*, pages 375–390, 2002.

